# Generalisation of A Class of Continuous Neural Networks

**John Shawe-Taylor**
Dept of Computer Science,
Royal Holloway, University of London,
Egham, Surrey TW20 0EX, UK
Email: john@dcs.rhbnc.ac.uk

**Jieyu Zhao**\*
IDSIA, Corso Elvezia 36,
6900-Lugano, Switzerland
Email: jieyu@carota.idsia.ch

## Abstract

We propose a way of using boolean circuits to perform real valued computation in a way that naturally extends their boolean functionality. The functionality of multiple fan in threshold gates in this model is shown to mimic that of a hardware implementation of continuous Neural Networks. A Vapnik-Chervonenkis dimension and sample size analysis for the systems is performed giving best known sample sizes for a real valued Neural Network. Experimental results confirm the conclusion that the sample sizes required for the networks are significantly smaller than for sigmoidal networks.

## 1 Introduction

Recent developments in complexity theory have addressed the question of complexity of computation over the real numbers. More recently attempts have been made to introduce some computational cost related to the accuracy of the computations [5]. The model proposed in this paper weakens the computational power still further by relying on classical boolean circuits to perform the computation using a simple encoding of the real values. Using this encoding we also show that $TC_0$ circuits interpreted in the model correspond to a Neural Network design referred to as Bit Stream Neural Networks, which have been developed for hardware implementation [8].

With the perspective afforded by the general approach considered here, we are also able to analyse the Bit Stream Neural Networks (or indeed any other adaptive system based on the technique), giving VC dimension and sample size bounds for PAC learning. The sample sizes obtained are very similar to those for threshold networks,

despite their being derived by very different techniques. They give the best bounds for neural networks involving smooth activation functions, being significantly lower than the bounds obtained recently for sigmoidal networks [4, 7].

We subsequently present simulation results showing that Bit Stream Neural Networks based on the technique can be used to solve a standard benchmark problem. The results of the simulations support the theoretical finding that for the same sample size generalisation will be better for the Bit Stream Neural Networks than for classical sigmoidal networks. It should also be stressed that the approach is very general – being applicable to any boolean circuit – and by its definition employs compact digital hardware. This fact motivates the introduction of the model, though it will not play an important part in this paper.

## 2   Definitions and Basic Results

A *boolean circuit* is a directed acyclic graph whose nodes are referred to as *gates*, with a single *output* node of out-degree zero. The nodes with in-degree zero are termed *input* nodes. The nodes that are not input nodes are *computational* nodes. There is a boolean function associated with each computational node of arity equal to its in-degree. The function computed by a boolean network is determined by assigning (input) values to its input nodes and performing the function at each computational node once its input values are determined. The result is the value at the output node. The class $TC_0$ is defined to be those functions that can be computed by a family of polynomially sized Boolean circuits with unrestricted fan-in and constant depth, where the gates are either NOT or THRESHOLD.

In order to use the boolean circuits to compute with real numbers we use the method of stochastic computing to encode real numbers as bit streams. The encoding we will use is to consider the stream of binary bits, for which the 1's are generated independently at random with probability $p$, as representing the number $p$. This is referred to as a Bernoulli sequence of probability $p$. In this representation, the multiplication of two independently generated streams can be achieved by a simple AND gate, since the probability of a 1 on the output stream is equal to $p_1 p_2$, where $p_1$ is the probability of a 1 on the first input stream and $p_2$ is the probability of a 1 on the second input stream. Hence, in this representation the boolean circuit consisting of a single AND gate can compute the product of its two inputs.

More background information about stochastic computing can be found in the work of Gaines [1]. The analysis we provide is made by treating the calculations as exact real valued computations. In a practical (hardware) implementation real bit streams would have to be generated [3] and the question of the accuracy of a delivered result arises.

In the applications considered here the output values are used to determine a binary value by comparing with a threshold of 0.5. Unless the actual output is exactly 1 or 0 (which can happen), then however many bits are collected at the output there is a slight probability that an incorrect classification will be made. Hence, the number of bits required is a function of the difference between the actual output and 0.5 and the level of confidence required in the correctness of the classification.

**Definition 1** *The real function computed by a boolean circuit $C$, which computes the boolean function*

$$f_C : \{0,1\}^n \longrightarrow \{0,1\},$$

*is the function*

$$g_C : [0,1]^n \longrightarrow [0,1],$$

*obtained by coding each input independently as a Bernoulli sequence and interpreting the output as a similar sequence.*

Hence, by the discussion above we have for the circuit $C$ consisting of a single AND gate, the function $g_C$ is given by $g_C(x_1, x_2) = x_1 x_2$.

We now give a proposition showing that the definition of real computation given above is well-defined and generalises the Boolean computation performed by the circuit.

**Proposition 2** *The bit stream on the output of a boolean circuit computing a real function is a Bernoulli sequence. The real function $g_C$ computed by an $n$ input boolean circuit $C$ can be expressed in terms of the corresponding boolean function $f_C$ as follows:*

$$g_C(x) = \sum_{\alpha \in \{0,1\}^n} P_x(\alpha) f_C(\alpha), \quad where \quad P_x(\alpha) = \prod_{i=1}^{n} x_i^{\alpha_i} (1 - x_i)^{(1-\alpha_i)}.$$

*In particular, $g_C|_{\{0,1\}^n} = f_C$.*

**Proof**: The output bit stream is a Bernoulli sequence, since the behaviour at each time step is independent of the behaviour at previous time sequences, assuming the input sequences are independent. Let the probability of a 1 in the output sequence be $p$. Hence, $g_C(x) = p$. At any given time the input to the circuit must be one of the $2^n$ possible binary vectors $\alpha$. $P_x(\alpha)$ gives the probability of the vector $\alpha$ occurring. Hence, the expected value of the output of the circuit is given in the proposition statement, but by the properties of a Bernoulli sequence this value is also $p$. The final claim holds since $P_\alpha(\alpha) = 1$, while $P_\alpha(\alpha') = 0$ for $\alpha \neq \alpha'$. ∎

Hence, the function computed by a circuit can be denoted by a polynomial of degree $n$, though the representation given above may involve exponentially many terms. This representation will therefore only be used for theoretical analysis.

## 3   Bit Stream Neural Networks

In this section we describe a neural network model based on stochastic computing and show that it corresponds to taking $TC_0$ circuits in the framework considered in Section 2.

A Stochastic Bit Stream Neuron is a processing unit which carries out very simple operations on its input bit streams. All input bit streams are combined with their corresponding weight bit streams and then the weighted bits are summed up. The final total is compared to a threshold value. If the sum is larger than the threshold the neuron gives an output 1, otherwise 0.

There are two different versions of the Stochastic Bit Stream Neuron corresponding to the different data representations. The definitions are given as follows.

**Definition 3 (AND-SBSN):** *A n-input AND version Stochastic Bit Stream Neuron has n weights in the range [-1,1] and n inputs in the range [0,1], which are all unipolar representations of Bernoulli sequences. An extra sign bit is attached to each weight Bernoulli sequence. The threshold $\theta$ is an integer lying between $-n$ to $n$ which is randomly generated according to the threshold probability density function $\phi(\theta)$. The computations performed during each operational cycle are*

*(1) combining respectively the n bits from n input Bernoulli sequences with the corresponding n bits from n weight Bernoulli sequences using the AND operation.*

*(2) assigning n weight sign bits to the corresponding output bits of the AND gate, summing up all the n signed output bits and then comparing the total with the randomly generated threshold value. If the total is not less than the threshold value, the AND-SBSN outputs 1, otherwise it outputs 0.*

We can now present the main result characterising the functionality of a Stochastic Bit Stream Neural Network as the real function of an $TC_0$ circuit.

**Theorem 4** *The functionality of a family of feedforward networks of Bit Stream Neurons with constant depth organised into layers with interconnections only between adjacent layers corresponds to the function $g_C$ for an $TC_0$ circuit $C$ of depth twice that of the network. The number of input streams is equal to the number of network inputs while the number of parameters is at most twice the number of weights.*

**Proof**: Consider first an individual neuron. We construct a circuit whose real functionality matches that of the neuron. The circuit has two layers. The first consists of a series of AND gates. Each gate links one input line of the neuron with its corresponding weight input. The outputs of these gates are linked into a threshold gate with fixed threshold $2d$ for the AND-SBSN, where $d$ is the number of input lines to the neuron. The threshold distribution of the AND SBSN is now simulated by having a series of $2d$ additional inputs to the threshold gate. The number of additional input streams required to simulate the threshold depends on how general a distribution is allowed for the threshold. We consider three cases:

1. If the threshold is fixed (i.e. not programmable), then no additional inputs are required, since the actual threshold can be suitably adapted.

2. If the threshold distribution is always focussed on one value (which can be varied), then an additional $\lceil \log_2(2d) \rceil$ ($\lceil \log_2(d) \rceil$) inputs are required to specify the binary value of this number. A circuit feeding the corresponding number of 1's to the threshold gate is not hard to construct.

3. In the fully general case any series of $2d + 1$ $(d + 1)$ numbers summing to one can be assigned as the probabilities of the possible values
$$\phi(0), \phi(1), \ldots, \phi(t),$$
where $t = 2d$ for the AND SBSN. We now construct a circuit which takes $t$ input streams and passes the 1-bits to the threshold gate of all the inputs up to the first input stream carrying a 0. No further input is passed to the threshold gate. In other words

    Threshold gate receives $s$ bits of input $\Leftrightarrow$ Input streams $1, \ldots, s$ have bit 1 and either $s = t$ or input stream $s + 1$ has input 0.

We now set the probability $p_s$ of stream $s$ as follows;
$$p_1 = 1 - \phi(0)$$
$$p_s = \frac{1 - \sum_{i=0}^{s-1} \phi(i)}{1 - \sum_{i=0}^{s-2} \phi(i)}$$
for $s = 2, \ldots, t$

With these values the probability of the threshold gate receiving $s$ bits is $\phi(s)$ as required.

This completes the replacement of a single neuron. Clearly, we can replace all neurons in a network in the same manner and construct a network with the required properties provided connections do not 'shortcut' layers, since this would create interactions between bits in different time slots. ∎

## 4   VC Dimension and Sample Sizes

In order to perform a VC Dimension and sample size analysis of the Bit Stream Neural Networks described in the previous section we introduce the following general framework.

**Definition 5** *For a set $\mathcal{G}$ of smooth functions $f : \mathcal{R}^n \times \mathcal{R}^\ell \to \mathcal{R}$, the class $\mathcal{F}$ is defined as*

$$\mathcal{F} = \mathcal{F}_\mathcal{G} = \{f_w | f_w(x) = f(x, w), f \in \mathcal{G}\}.$$

*The corresponding classification class obtained by taking a fixed set of $s$ of the functions from $\mathcal{G}$, thresholding the corresponding functions from $\mathcal{F}$ at 0 and combining them (with the same parameter vector) in some logical formula will be denoted $H_s(\mathcal{F})$. We will denote $H_1(\mathcal{F})$ by $H(\mathcal{F})$.*

In our case we will consider a set of circuits $\mathcal{C}$ each with $n + \ell$ input connections, $n$ labelled as the input vector and $\ell$ identified as parameter input connections. Note that if circuits have too few input connections, we can pad them with dummy ones. The set $\mathcal{G}$ will then be the set

$$\mathcal{G} = \mathcal{G}_\mathcal{C} = \{g_C | C \in \mathcal{C}\},$$

while $\mathcal{F}_{\mathcal{G}_\mathcal{C}}$ will be denoted by $\mathcal{F}_\mathcal{C}$.

We now quote some of the results of [7] which uses the techniques of Karpinski and MacIntyre [4] to derive sample sizes for classes of smoothly parametrised functions.

**Proposition 6** *[7] Let $\mathcal{G}$ be the set of polynomials $p$ of degree at most $d$ with $p : \mathcal{R}^n \times \mathcal{R}^\ell \to \mathcal{R}$ and*

$$\mathcal{F} = \mathcal{F}_\mathcal{G} = \{p_w | p_w(x) = p(x, w), p \in \mathcal{G}\}.$$

*Hence, there are $\ell$ adjustable parameters and the input dimension is $n$. Then the VC-dimension of the class $H_s(\mathcal{F}_\mathcal{C})$ is bounded above by*

$$\log_2(2(2d)^\ell) + 17\ell \log_2(s).$$

**Corollary 7** *For a set of circuits $\mathcal{C}$, with $n$ input connections and $\ell$ parameter connections, the VC-dimension of the class $H_s(\mathcal{F}_\mathcal{C})$ is bounded above by*

$$1 + \ell(1 + \log_2(n + \ell) + 17 \log_2(s)).$$

**Proof**: By Proposition 2 the function $g_C$ computed by a circuit $C$ with $t$ input connections has the form

$$g_C(x) = \sum_{\alpha \in \{0,1\}^t} P_x(\alpha) f_C(\alpha), \quad \text{where} \quad P_x(\alpha) = \prod_{i=1}^t x_i^{\alpha_i} (1 - x_i)^{(1 - \alpha_i)}.$$

Hence, $g_C(x)$ is a polynomial of degree $t$. In the case considered the number $t$ of input connections is $n + \ell$. The result follows from the proposition. ∎

**Proposition 8** *[7] Let $\mathcal{G}$ be the set of polynomials $p$ of degree at most $d$ with $p: \mathcal{R}^n \times \mathcal{R}^\ell \to \mathcal{R}$ and*

$$\mathcal{F} = \mathcal{F}_\mathcal{G} = \{p_w | p_w(x) = p(x,w), p \in \mathcal{G}\}.$$

*Hence, there are $\ell$ adjustable parameters and the input dimension is $n$. If a function $h \in H_s(\mathcal{F})$ correctly computes a function on a sample of $m$ inputs drawn independently according to a fixed probability distribution, where*

$$m \geq m_0(\epsilon, \delta) = \frac{1}{\epsilon(1-\sqrt{\epsilon})}\left[2\ell\ln\left(\frac{4e\sqrt{sd}}{\epsilon}\right) + \ln\left(\frac{2\ell/(\ell-1)}{\delta}\right)\right]$$

*then with probability at least $1 - \delta$ the error rate of $h$ will be less than $\epsilon$ on inputs drawn according to the same distribution.*

**Corollary 9** *For a set of circuits $\mathcal{C}$, with $n$ input connections and $\ell$ parameter connections, If a function $h \in H_s(\mathcal{F}_\mathcal{C})$ correctly computes a function on a sample of $m$ inputs drawn independently according to a fixed probability distribution, where*

$$m \geq m_0(\epsilon, \delta) = \frac{1}{\epsilon(1-\sqrt{\epsilon})}\left[2\ell\ln\left(\frac{4e\sqrt{s(n+\ell)}}{\epsilon}\right) + \ln\left(\frac{2\ell/(\ell-1)}{\delta}\right)\right]$$

*then with probability at least $1 - \delta$ the error rate of $h$ will be less than $\epsilon$ on inputs drawn according to the same distribution.*

**Proof**: As in the proof of the previous corollary, we need only observe that the functions $g_C$ for $C \in \mathcal{C}$ are polynomials of degree at most $n + \ell$. ∎

Note that the best known sample sizes for threshold networks are given in [6]:

$$m \geq m_0(\epsilon, \delta) = \frac{1}{\epsilon(1-\sqrt{\epsilon})}\left[2W\ln\left(\frac{6\sqrt{N}}{\epsilon}\right) + \ln\left(\frac{\ell/(\ell-1)}{\delta}\right)\right],$$

where $W$ is the number of adaptable weights (parameters) and $N$ is the number of computational nodes in the network. Hence, the bounds given above are almost identical to those for threshold networks, despite the underlying techniques used to derive them being entirely different.

One surprising fact about the above results is that the VC dimension and sample sizes are independent of the complexity of the circuit (except in as much as it must have the required number of inputs). Hence, additional layers of fixed computation cannot increase the sample complexity above the bound given).

## 5   Simulation Results

The Monk's problems which were the basis of a first international comparison of learning algorithms, are derived from a domain in which each training example is represented by six discrete-valued attributes. Each problem involves learning a binary function defined over this domain, from a sample of training examples of this function. The 'true' concepts underlying each Monk's problem are given by:

MONK-1: ($attribute_1 = attribute_2$)
        or ($attribute_5 = 1$)
MONK-2: ($attribute_i = 1$)
        for EXACTLY TWO $i \in \{1, 2, ..., 6\}$
MONK-3: ($attribute_5 = 3$ *and* $attribute_4 = 1$)
        or ($attribute_5 \neq 4$ *and* $attribute_2 \neq 3$)

There are 124, 169 and 122 samples in the training sets of MONK-1, MONK-2 and MONK-3 respectively. The testing set has 432 patterns. The network had 17 input units, 10 hidden units, 1 output unit, and was fully connected. Two networks were used for each problem. The first was a standard multi-layer perceptron with sigmoid activation function trained using the backpropagation algorithm (BP Network).

The second network had the same architecture, but used bit stream neurons in place of sigmoid ones (BSN Network). The functionality of the neurons was simulated using probability generating functions to compute the probability values of the bit streams output at each neuron. The backpropagation algorithm was adapted to train these networks by computing the derivative of the output probability value with respect to the individual inputs to that neuron [8].

Experiments were performed with and without noise in the training examples. There is 5% additional noise (misclassifications) in the training set of MONK-3. The results for the Monk's problems using the moment generating function simulation are shown as follows:

|        | BP Network | | BSN Network | |
|--------|----------|---------|----------|---------|
|        | training | testing | training | testing |
| MONK-1 | 100%     | 86.6%   | 100%     | 97.7%   |
| MONK-2 | 100%     | 84.2%   | 100%     | 100%    |
| MONK-3 | 97.1%    | 83.3%   | 98.4%    | 98.6%   |

It can be seen that the generalisation of the BSN network is much better than that of a general multilayer backpropagation network. The results on MONK-3 problem is extremely good. The results reported by Hassibi and Stork [2] using a sophisticated weight pruning technique are only 93.4% correct for the training set and 97.2% correct for the testing set.

## Footnotes

\*Work performed while at Royal Holloway, University of London

## References

[1] B. R. Gaines, Stochastic Computing Systems, Advances in Information Systems Science 2 (1969) pp37-172.

[2] B. Hassibi and D.G. Stork, Second order derivatives for network pruning: Optimal brain surgeon, Advances in Neural Information Processing System, Vol 5 (1993) 164–171.

[3] P. Jeavons, D.A. Cohen and J. Shawe-Taylor, Generating Binary Sequences for Stochastic Computing, IEEE Trans on Information Theory, 40 (3) (1994) 716–720.

[4] M. Karpinski and A. MacIntyre, Bounding VC-Dimension for Neural Networks: Progress and Prospects, Proceedings of EuroCOLT'95, 1995, pp. 337–341, Springer Lecture Notes in Artificial Intelligence, 904.

[5] P. Koiran, A Weak Version of the Blum, Shub and Smale Model, ESPRIT Working Group NeuroCOLT Technical Report Series, NC-TR-94-5, 1994.

[6] J. Shawe-Taylor, Threshold Network Learning in the Presence of Equivalences, Proceedings of NIPS 4, 1991, pp. 879–886.

[7] J. Shawe-Taylor, Sample Sizes for Sigmoidal Networks, to appear in the Proceedings of Eighth Conference on Computational Learning Theory, COLT'95, 1995.

[8] John Shawe-Taylor, Peter Jeavons and Max van Daalen, "Probabilistic Bit Stream Neural Chip : Theory", Connection Science, Vol 3, No 3, 1991.
